# Efficient Bregman Range Search

**Lawrence Cayton**
Max Planck Institute for Biological Cybernetics
lcayton@tuebingen.mpg.de

## Abstract

We develop an algorithm for efficient range search when the notion of dissimilarity is given by a Bregman divergence. The range search task is to return all points in a potentially large database that are within some specified distance of a query. It arises in many learning algorithms such as locally-weighted regression, kernel density estimation, neighborhood graph-based algorithms, and in tasks like outlier detection and information retrieval. In metric spaces, efficient range search-like algorithms based on spatial data structures have been deployed on a variety of statistical tasks. Here we describe an algorithm for range search for an arbitrary Bregman divergence. This broad class of dissimilarity measures includes the relative entropy, Mahalanobis distance, Itakura-Saito divergence, and a variety of matrix divergences. Metric methods cannot be directly applied since Bregman divergences do not in general satisfy the triangle inequality. We derive geometric properties of Bregman divergences that yield an efficient algorithm for range search based on a recently proposed space decomposition for Bregman divergences.

## 1 Introduction

Range search is a fundamental proximity task at the core of many learning problems. The task of range search is to return all points in a database within a specified distance of a given query. The problem is to do so efficiently, without examining the entire database. Many machine learning algorithms require range search. Locally weighted regression and kernel density estimation/regression both require retrieving points in a region around a test point. Neighborhood graphs—used in manifold learning, spectral algorithms, semisupervised algorithms, and elsewhere—can be built by connecting each point to all other points within a certain radius; doing so requires range search at each point. Computing point-correlation statistics, distance-based outliers/anomalies, and intrinsic dimensionality estimates also requires range search.

A growing body of work uses spatial data structures to accelerate the computation of these and other proximity problems for statistical tasks. This line of techniques, coined "$n$-body methods" in [11], has showed impressive speedups on a variety of tasks including density estimation [12], gaussian process regression [25], non-parametric classification [17], matrix approximation [14], and kernel summation [15]. These methods achieve speedups by pruning out large portions of the search space with bounds derived from KD or metric trees that are augmented with statistics of the database. Some of these algorithms are direct applications of range search; others rely on very similar pruning techniques. One fairly substantial limitation of these methods is that they all derive bounds from the triangle inequality and thus only work for notions of distance that are metrics.

The present work is on performing range search efficiently when the notion of dissimilarity is not a metric, but a *Bregman divergence*. The family of Bregman divergences includes the standard $\ell_2^2$ distance, Mahalanobis distance, KL-divergence, Itakura-Saito divergence, and a variety of matrix dissimilarity measures. We are particularly interested in the KL-divergence, as it is not a metric and is used extensively in machine learning. It appears naturally in document analysis, since documents

are often modeled using histograms [22, 5]. It also is used in many vision applications [23], such as content-based image retrieval [24]. Because Bregman divergences can be asymmetric and need not satisfy the triangle inequality, the traditional metric methods cannot be applied.

In this work we present an algorithm for efficient range search when the notion of dissimilarity is an arbitrary Bregman divergence. These results demonstrate that the basic techniques behind the previously described efficient statistical algorithms can be applied to non-metric dissimilarities including, notably, the KL-divergence. Because of the widespread use of histogram representations, this generalization is important.

The task of efficient Bregman range search presents a technical challenge. Our algorithm cannot rely on the triangle inequality, so bounds must be derived from geometric properties of Bregman divergences. The algorithm makes use of a simple space decomposition scheme based on *Bregman balls* [8], but deploying this decomposition for the range search problem is not straightforward. In particular, one of the bounds required results in a *non*-convex program to be solved, and the other requires comparing two convex bodies. We derive properties of Bregman divergences that imply efficient algorithms for these problems.

## 2 Background

In this section, we briefly review prior work on Bregman divergences and proximity search. Bregman divergences originate in [7] and have become common in the machine learning literature, *e.g.* [3, 4].

**Definition 1.** *Let $f : \mathbb{R}^D \to \mathbb{R}$ be strictly convex and differentiable. The Bregman divergence based on $f$ is*

$$d_f(x, y) \equiv f(x) - f(y) - \langle \nabla f(y), x - y \rangle.$$

As can be seen from the definition, a Bregman divergence measures the distance between a function and its first-order taylor series approximation. Standard examples include $f(x) = \frac{1}{2}\|x\|_2^2$, yielding the $\ell_2^2$ distance $d_f(x, y) = \frac{1}{2}\|x - y\|_2^2$, and $f(x) = \sum_i x_i \log x_i$, giving the KL-divergence $d_f(x, y) = \sum_i x_i \log \frac{x_i}{y_i}$ The Itakura-Saito divergence and Mahalanobis distance are other examples of Bregman divergences.

Strict convexity of $f$ implies that $d_f(x, y) \geq 0$, with equality if, and only if, $x = y$. Though Bregman divergences satisfy this non-negativity property, like metrics, the similarities to metrics end there. In particular, a Bregman divergence need not satisfy the triangle inequality or be symmetric.

Bregman divergences do possess several geometric properties related to the convexity of the base function. Most notably, $d_f(x, y)$ is always convex in $x$ (though not necessarily in $y$), implying that the *Bregman ball*

$$B_f(\mu, R) \equiv \{x \mid d_f(x, \mu) \leq R\}$$

is a convex body.

Recently, work on a variety of geometric tasks with Bregman divergences has appeared. In [19], geometric properties of Bregman voronoi diagrams are derived. [1] studies core-sets under Bregman divergences and gives a provably correct approximation algorithm for $k$-median clustering. [13] examines sketching Bregman (and Csiszár) divergences. [8] describes the Bregman ball tree in the context of nearest neighbor search; we will describe this work further momentarily. As these papers demonstrate, there has been substantial recent interest in developing basic geometric algorithms for Bregman divergences. The present paper contributes an effective algorithm for range search, one of the core problems of computational geometry [2], to this repertoire.

The Bregman ball tree (BB-tree) was introduced in the context of nearest neighbor (NN) search [8]. Though NN search has a similar flavor to range search, the bounds that suffice for NN search are not sufficient for range search. Thus the utility of the BB-tree for statistical tasks is at present rather seriously limited. Moreover, though the extension of metric trees to range search (and hence to the previously described statistical tasks) is fairly straightforward because of the triangle inequality, the extension of BB-trees is substantially more complex.

Several other papers on Bregman proximity search have appeared very recently. Nielsen *et al.* study some improvements to the BB-tree [21] and develop a related data structure which can be used with symmetrized divergences [20]. Zhang *et al.* develop extensions of the VA-file and the R-tree for Bregman divergences [26]. These data structures can be adapted to work for Bregman divergences, as the authors of [26] demonstrate, because bounds on the divergence from a query to a rectangular cell can be computed cheaply; however this idea appears limited to *decomposable* Bregman divergences—divergences that decompose into a sum over one-dimensional divergences.[1] Nevertheless, these data structures seem practical and effective and it would be interesting to apply them to statistical tasks.[2] The applicability of rectangular cell bounds was independently demonstrated in [9, Chapter 7], where it is mentioned that KD-trees (and relatives) can be used for decomposable Bregman divergences. That chapter also contains theoretical results on the general Bregman range search problem attained by adapting known data structures via the lifting technique (also used in [26] and previously in [19]).

# 3 Range search with BB-trees

In this section, we review the Bregman ball tree data structure and outline the range search algorithm. The search algorithm relies on geometric properties of Bregman divergences, which we derive in section 4.

The BB-tree is a hierarchical space decomposition based on Bregman balls. It is a binary tree defined over the database such that each level provides a partition of the database points. As the tree is descended, the partition becomes finer and finer. Each node $i$ in the tree owns a subset of the points $X_i$ and also defines a Bregman ball $B_f(\mu, R)$ such that $X_i \subset B_f(\mu, R)$. If $i$ is an interior node, it has two children $j$ and $k$ that encapsulate database points $X_j$ and $X_k$. Moreover, each point in $X_i$ is in exactly one of $X_j$ and $X_k$. Each leaf node contains some small number of points and the root node contains the entire database.

Here we use this simple form of BB-tree, though our results apply to any hierarchical space decomposition based on Bregman balls, such as the more complex tree described in [21].

To encourage a rapid rate of radius decrease, an effective build algorithm will split a node into two well-separated and compact children. Thus a reasonable method for building BB-trees is to perform a top-down hierarchical clustering. Since $k$-means has been generalized to arbitrary Bregman divergences [4], it is a natural choice for a clustering algorithm.

## 3.1 Search algorithm

We now turn to the search algorithm, which uses a branch-and-bound approach. We develop the necessary novel bounding techniques in the next section.

Suppose we are interested in returning all points within distance $\gamma$ of a query $q$—*i.e.* we hope to retrieve all database points lying inside of $B_q \equiv B_f(q, \gamma)$. The search algorithm starts at the root node and recursively explores the tree. At a node $i$, the algorithm compares the node's Bregman ball $B_x$ to $B_q$. There are three possible situations. First, if $B_x$ is contained in $B_q$, then all $x \in B_x$ are in the range of interest. We can thus stop the recursion and return all the points associated with the node without explicitly computing the divergence to any of them. This type of pruning is called *inclusion* pruning. Second, if $B_x \cap B_q = \emptyset$, the algorithm can prune out $B_x$ and stop the recursion; none of these points are in range. This is *exclusion* pruning. See figure 1. All performance gains from using the algorithm come from these two types of pruning. The third situation is $B_x \cap B_q \neq \emptyset$ and $B_x \not\subset B_q$. In this situation, the algorithm cannot perform any pruning, so recurses on the children of node $i$. If $i$ is a leaf node, then the algorithm computes the divergence to each database point associated with $i$ and returns those elements within range.

The two types of pruning—inclusion and exclusion—have been applied to a variety of problems with metric and KD-trees, see *e.g.* [11, 12, 25] and the papers cited previously. Thus though we

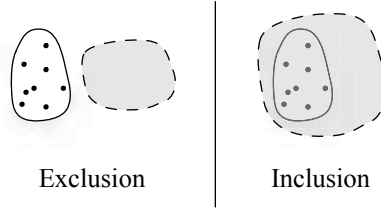

Exclusion | Inclusion

Figure 1: The two pruning scenarios. The dotted, shaded object is the query range and the other is the Bregman ball associated with a node of the BB-tree.

focus on range search, these types of prunings are useful in a broad range of statistical problems. A third type of pruning, *approximation* pruning, is useful in tasks like kernel density estimation [12]. This type of pruning is another form of inclusion pruning and can be accomplished with the same technique.

It has been widely observed that the performance of spatial decomposition data structures, degrades with increasing dimensionality. In order to manage high-dimensional datasets, practitioners often use *approximate* proximity search techniques [8, 10, 17]. In the experiments, we explore one way to use the BB-tree in an approximate fashion.

Determining whether two Bregman balls intersect, or whether one Bregman ball contains another, is non-trivial. For the range search algorithm to be effective, it must be able to determine these relationships very quickly. In the case of metric balls, these determinations are trivially accomplished using the triangle inequality. Since we cannot rely on the triangle inequality for an arbitrary Bregman divergence, we must develop novel techniques.

## 4 Computation of ball intersection

In this section we lay out the main technical contribution of the paper. We develop algorithms for determining (1) whether one Bregman ball is contained in another and (2) whether two Bregman balls have non-empty intersection.

### 4.1 Containment

Let $B_q$ $B_f(\ _q\ R_q)$ and $B_x$ $B_f(\ _x\ R_x)$. We wish to evaluate if $B_x$ $B_q$. This problem is equivalent to testing whether

$$d_f(x\quad _q)\quad R_q$$

for all $x$ $B_x$. Simplifying notation, the core problem is determining

$$\max_x\ d_f(x\ q)$$
$$\text{subject to: } d_f(x\quad)\quad R \tag{maxP}$$

Unfortunately, this problem is not convex. As is well-known, non-convex problems are in general much more computationally difficult to solve than convex ones. This difficulty is particularly problematic in the case of range search, as the search algorithm will need to solve this problem repeatedly in the course of evaluating a singe range query. Moreover, finding a sub-optimal solution (*i.e.* a point $x$ $B_f(\ R)$ that is not the max) will render the solution to the range search incorrect.

Remarkably, beneath (maxP) lies a geometric structure that allows an efficient solution. We now show the main claim of this section, which implies a simple, efficient algorithm for solving (maxP). We denote the convex conjugate of $f$ by

$$f\ (x)\quad \sup_y\ x\ y\subset\ f(y)$$

and define $x$ $f(x), q$ $f(q)$, *etc.*

**Claim 1.** *Suppose that the domain of $f$ is $C$ and that $B_f(\mu, R) \subset \text{relint}(C)$. Furthermore, assume that $\|\nabla^2 f^*(x')\|$ is lower-bounded for all $x'$ such that $x \in B_f(\mu, R)$. Let $x_p$ be the optimal solution to (maxP). Then $x'_p$ lies in the set $\{\theta\mu' + (1-\theta)q' \mid \theta \geq 0\}$.*

*Proof.* Though the program is not concave, the Lagrange dual still provides an upper bound on the optimal solution value (by weak duality). The Lagrangian is

$$\nu(x, \lambda) \equiv d_f(x, q) - \lambda(d_f(x, \mu) - R), \tag{1}$$

where $\lambda \geq 0$.

Differentiating (1) with respect to $x$ and setting it equal to 0, we get

$$\nabla f(x_p) - \nabla f(q) - \lambda \nabla f(x_p) + \lambda \nabla f(\mu) = 0,$$

which implies that

$$\nabla f(x_p) = \frac{1}{1-\lambda}\left(\nabla f(q) - \lambda \nabla f(\mu)\right). \tag{2}$$

We need to check what type of extrema $\nabla f(x_p) = 0$ is:

$$\nabla_x^2 \nu(x, \lambda) = (1-\lambda)\nabla^2 f(x).$$

Thus for $\lambda > 1$, the $x_p$ defined implicitly in (2) is a maximum. Setting $\theta \equiv -\frac{\lambda}{1-\lambda}$ gives

$$\nabla f(x_p) = \theta\mu' + (1-\theta)q',$$

where $\theta \in (-\infty, 0) \cup (1, \infty)$; we restrict attention to $\theta \in (1, \infty)$ since that is where $\lambda > 1$ and hence $x_p$ is a maximum. Let $x'_\theta \equiv \theta\mu' + (1-\theta)q'$ and $x_\theta \equiv \nabla f^*(x'_\theta)$. The Lagrange dual is

$$\mathcal{L}(\theta) \equiv d_f(x_\theta, q) + \frac{\theta}{1-\theta}(d_f(x_\theta, \mu) - R).$$

Then for any $\theta \in (1, \infty)$, we have

$$d_f(x_p, q) \leq \mathcal{L}(\theta) \tag{3}$$

by weak duality. We now show that there is a $\theta^* > 1$ satisfying $d_f(x_{\theta^*}, \mu) = R$. One can check that the derivative of $d_f(x_\theta, \mu)$ with respect to $\theta$ is

$$(\theta - 1)(\mu' - q')^\top \nabla^2 f^*(x'_\theta)(\mu' - q'). \tag{4}$$

Since $\|\nabla^2 f^*\| > c$, for some positive $c$, (4) is at least $(\theta - 1)\|\mu' - q'\|c$. We conclude that $d_f(x_\theta, \mu)$ is increasing at an increasing rate with $\theta$. Thus there must be some $\theta^* > 1$ such that $d_f(x_{\theta^*}, \mu) = R$. Plugging this $\theta^*$ into the dual, we get

$$\mathcal{L}(\theta^*) = d_f(x_{\theta^*}, q) + \frac{\theta^*}{1-\theta^*}(d_f(x_{\theta^*}, \mu) - R)$$

$$= d_f(x_{\theta^*}, q).$$

Combining with (3), we have

$$d_f(x_p, q) \leq d_f(x_{\theta^*}, \mu).$$

Finally, since (maxP) is a maximization problem and since $x_{\theta^*}$ is feasible, the previous inequality is actually an equality, giving the theorem. $\square$

Thus determining if $B_x \subset B_q$ reduces to searching for $\theta^* > 1$ satisfying

$$d_f(x_{\theta^*}, \mu_x) = R_x$$

and comparing $d_f(x_{\theta^*}, \mu_q)$ to $R_q$. Note that there is no obvious upper bound on $\theta^*$ in general, though one may be able to derive such a bound for a particular Bregman divergence. Without such an upper bound, one needs to use a line search method that does not require one, such as Newton's method or the secant method. Both of these line search methods will converge quickly (quadratic in the case of Newton's method, slightly slower in the case of the secant method): since $d_f(x_\theta, \mu_x)$ is monotonic in $\theta$, there is a unique root.

Interestingly, the *convex* program evaluated in [8] has a similar solution space, which we will again encounter in the next section.

## 4.2 Non-empty intersection

In this section we provide an algorithm for evaluating whether $B_q \cap B_x = \emptyset$. We will need to make use of the *Pythagorean theorem*, a standard property of Bregman divergences.

**Theorem 1** (Pythagorean). *Let $C \subset \mathbb{R}^D$ be a convex set and let $x \in C$. Then for all $z$, we have*

$$d_f(x, z) \geq d_f(x, y) + d_f(y, z),$$

*where $y \equiv \mathrm{argmin}_{y \in C} d_f(y, z)$ is the projection of $z$ onto $C$.*

At first glance, the Pythagorean theorem may appear to be a triangle inequality for Bregman divergences. However, the inequality is actually the *reverse* of the standard triangle inequality and only applies to the very special case when $y$ is the projection of $z$ onto a convex set containing $x$. We now prove the main claim of this section.

**Claim 2.** *Suppose that $B_x \cap B_q \neq \emptyset$. Then there exists a $w$ in*

$$\{\nabla f^*(\theta \mu'_x + (1 - \theta)\mu'_q) \mid \theta \in [0, 1]\}$$

*such that $w \in B_q \cap B_x$.*

*Proof.* Let $z \in B_x \cap B_q$. We will refer to the set $\{\nabla f^*(\theta \mu'_x + (1 - \theta)\mu'_q) \mid \theta \in [0, 1]\}$ as the *dual curve*.

Let $x$ be the projection of $\mu_q$ onto $B_x$ and let $q$ be the projection of $\mu_x$ onto $B_q$. Both $x$ and $q$ are on the dual curve (this fact follows from [8, Claim 2]), so we are done if we can show that at least one of them lies in the intersection of $B_x$ and $B_q$. Suppose towards contradiction that neither are in the intersection.

The projection of $x$ onto $B_q$ lies on the dual curve between $x$ and $\mu_y$; thus projecting $x$ onto $B_q$ yields $q$ and similarly projecting $q$ onto $B_x$ yields $x$. By the Pythagorean theorem,

$$d_f(z, x) \geq d_f(z, q) + d_f(q, x), \tag{5}$$

since $q$ is the projection of $x$ onto $B_q$ and since $z \in B_q$. Similarly,

$$d_f(z, q) \geq d_f(z, x) + d_f(x, q). \tag{6}$$

Inserting (5) into (6), we get

$$d_f(z, q) \geq d_f(z, q) + d_f(q, x) + d_f(x, q).$$

Rearranging, we get that $d_f(q, x) + d_f(x, q) \leq 0$. Thus both $d_f(q, x) = 0$ and $d_f(x, q) = 0$, implying that $x = q$. But since $x \in B_x$ and $q \in B_q$, we have that $x = q \in B_q \cap B_q$. This is the desired contradiction. $\square$

The proceeding claim yields a simple algorithm for determining whether two balls $B_x$ and $B_q$ are disjoint: project $\mu_x$ onto $B_q$ using the line search algorithm discussed previously. The projected point will obviously be in $B_q$; if it is also in $B_x$, the two balls intersect.[3] Otherwise, they are disjoint and exclusion pruning can be performed.

# 5 Experiments

We compare the performance of the search algorithm to standard brute force search on several datasets. We are particularly interested in text applications as histogram representations are common, datasets are often very large, and efficient search is broadly useful. We experimented with the following datasets, many of which are fairly high-dimensional.

- **pubmed-$D$.** We used one million documents from the pubmed abstract corpus (available from the UCI collection). We generated a *correlated topic model* (CTM) [5] with $D = 4, 8, \ldots, 256$ topics. For each $D$, we built a CTM using a training set and then performed inference on the 1M documents to generate the topic histograms.

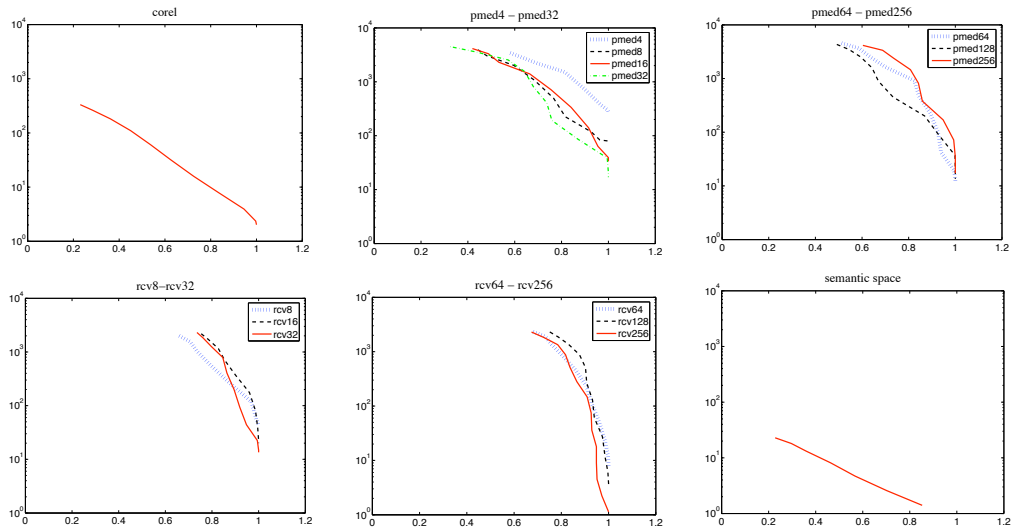

Figure 2: Approximate search. The $y$-axis is on a logarithmic scale and is the speedup over brute force search. The $x$ axis is a linear scale and is the average percentage of the points in range returned (*i.e.* the average recall).

- **Corel histograms.** This data set consists of 60k color histograms of dimensionality 64 generated from the Corel image datasets.

- **rcv-$D$.** Latent dirichlet allocation was applied to 500K documents from the rcv1 [16] corpus to generate topic histograms for each [6]. $D$ is set to $8, 16, 32, \ldots 256$.

- **Semantic space**. This dataset is a 371-dimensional representation of 5000 images from the Corel stock photo collection. Each image is represented as a distribution over keywords [24].

All of our experiments are for the KL-divergence. Although the KL-divergence is widely used, little is known about efficient proximity techniques for it. In contrast, the $\ell_2^2$ and Mahalanobis distances can be handled by metric methods, for which there is a huge literature. Application of the range search algorithm for the KL-divergence raises one technical point: Claim 1 requires that the KL-ball being investigated lies within the domain of the KL-divergence. It is possible that the ball will cross the domain boundary ($x_i = 0$), though we found that this was not a significant issue. When it did occur (which can be checked by evaluating $d_f(\mu, x_\theta)$ for large $\theta$), we simply did not perform inclusion pruning for that node.

There are two regimes where range search is particularly useful: when the radius $\gamma$ is very small and when it is large. When $\gamma$ is small, range search is useful in instance-based learning algorithms like locally weighted regression, which need to retrieve points close to each test point. It is also useful for generating neighborhood graphs. When $\gamma$ is large enough that $B_f(q, \gamma)$ will contain most of the database, range search is potentially useful for applications like distance-based outlier detection and anomaly detection. We provide experiments for both of these regimes.

Table 1 shows the results for exact range search. For the small radius experiments, $\gamma$ was chosen so that about 20 points would be inside the query ball (on average). On the pubmed datasets, we are getting one to two orders of magnitude speed-up across all dimensionalities. On the rcv datasets, the BB-tree range search algorithm is an order of magnitude faster than brute search except of the the two datasets of highest dimensionality. The algorithm provides a useful speedup on corel, but no speedup on semantic space. We note that the semantic space dataset is both high-dimensional (371 dimensions) and quite small (5k), which makes it very hard for proximity search. The algorithm reflects the widely observed phenomenon that the performance of spatial decomposition data structures degrades with dimensionality, but still provides a useful speedup on several moderate-dimensional datasets.

Table 1: Exact range search.

| dataset | dimensionality | speedup | |
| --- | --- | --- | --- |
| | | small radius | large radius |
| corel | 64 | 2.53 | 3.4 |
| pubmed4 | 4 | 371.6 | 5.1 |
| pubmed8 | 8 | 102.7 | 9.7 |
| pubmed16 | 16 | 37.3 | 12.8 |
| pubmed32 | 32 | 18.6 | 47.1 |
| pubmed64 | 64 | 13.26 | 21.6 |
| pubmed128 | 128 | 15.0 | 120.4 |
| pubmed256 | 256 | 18.9 | 39.0 |
| rcv8 | 8 | 48.1 | 8.9 |
| rcv16 | 16 | 23.0 | 21.9 |
| rcv32 | 32 | 16.4 | 16.4 |
| rcv64 | 64 | 11.4 | 9.6 |
| rcv128 | 128 | 6.1 | 3.1 |
| rcv256 | 256 | 1.1 | 1.9 |
| semantic space | 371 | .7 | 1.0 |

For the large radius experiments, $\gamma$ was chosen so that all but about 100-300 points would be in range. The results here are more varied than for small $\gamma$, but we are still getting useful speedups across most of the datasets. Interestingly, the amount of speedup seems less dependent of the dimensionality in comparison to the small $\gamma$ experiments.

Finally, we investigate approximate search, which we consider the most likely use of this algorithm. There are many ways to use the BB-tree in an approximate way. Here, we follow [18] and simply cut-off the search process early. We are thus guaranteed to get only points within the specified range (perfect precision), but we may not get all of them (less than perfect recall). In instance-based learning algorithms, this loss of recall is often tolerable as long as a reasonable number of points are returned. Thus a practical way to deploy the range search algorithm is to run it until enough points are recovered. In this experiment, $\gamma$ was set so that about 50 points would be returned. Figure 2 shows the results.

These are likely the most relevant results to practical applications. They demonstrate that the proposed algorithm provides a speedup of up to four orders of magnitude with a high recall.

## 6   Conclusion

We presented the first algorithm for efficient ball range search when the notion of dissimilarity is an arbitrary Bregman divergence. This is an important step towards generalizing the efficient proximity algorithms from $\ell_2$ (and metrics) to the family of Bregman divergences, but there is plenty more to do. First, it would be interesting to see if the *dual-tree* approach promoted in [11, 12] and elsewhere can be used with BB-trees. This generalization appears to require more complex bounding techniques than those discussed here. A different research goal is to develop efficient algorithms for proximity search that have rigorous guarantees on run-time; theoretical questions about proximity search with Bregman divergences remain largely open. Finally, the work in this paper provides a foundation for developing efficient statistical algorithms using Bregman divergences; fleshing out the details for a particular application is an interesting direction for future research.

## Footnotes

[1]This assumption is implicit in the proof of [26, Lemma 3.1] and is used in the revised lower bound computation as well.

[2][26] had yet not been published at the time of submission of the present work and hence we have not yet done a detailed comparison.

[3]Claim 2 actually only shows that at least one of two projections—$\mu_x$ onto $B_q$ and $\mu_q$ onto $B_x$—will be in the intersection. However, one can show that *both* projections will be in the intersection using the monotonicity of $d_f(x_\theta, \cdot)$ in $\theta$.

## References

[1] Marcel Ackermann and Johannes Blömer. Coresets and approximate clustering for bregman divergences. In *Proceedings of the Symposium on Discrete Algorithms (SODA)*, 2009.

[2] Pankaj K. Agarwal and Jeff Erickson. Geometric range searching and its relatives. In *Advances in Discrete and Computational Geometry*, pages 1–56. American Mathematical Society, 1999.

[3] Katy Azoury and Manfred Warmuth. Relative loss bounds for on-line density estimation with the exponential family of distributions. *Machine Learning*, 43(3):211–246, 2001.

[4] Arindam Banerjee, Srujana Merugu, Inderjit S. Dhillon, and Joydeep Ghosh. Clustering with bregman divergences. *Journal of Machine Learning Research*, Oct 2005.

[5] David Blei and John Lafferty. A correlated topic model of *Science*. *Annals of Applied Statistics*, 1(1):17–35, 2007.

[6] David Blei, Andrew Ng, and Michael Jordan. Latent dirichlet allocation. *Journal of Machine Learning Research*, 2003.

[7] L.M. Bregman. The relaxation method of finding the common point of convex sets and its application to the solution of problems in convex programming. *USSR Computational Mathematics and Mathematical Physics*, 7(3):200–217, 1967.

[8] Lawrence Cayton. Fast nearest neighbor retrieval for bregman divergences. In *Proceedings of the International Conference on Machine Learning*, 2008.

[9] Lawrence Cayton. *Bregman Proximity Search*. PhD thesis, University of California, San Diego, 2009.

[10] Mayur Datar, Nicole Immorlica, Piotr Indyk, and Vahab S. Mirrokni. Locality-sensitive hashing scheme based on p-stable distributions. In *Symposium on Computational Geometry*, 2004.

[11] Alexander Gray and Andrew Moore. 'N-body' problems in statistical learning. In *Advances in Neural Information Processing Systems*, 2000.

[12] Alexander Gray and Andrew Moore. Nonparametric density estimation: Toward computational tractability. In *SIAM International Conference on Data Mining*, 2003.

[13] Sudipto Guha, Piotr Indyk, and Andrew McGregor. Sketching information divergences. In *Conference on Learning Theory*, 2007.

[14] Michael P. Holmes, Alexander Gray, and Charles Lee Isbell. QUIC-SVD: Fast SVD using cosine trees. In *Advances in Neural Information Processing Systems 21*, 2008.

[15] Dongryeol Lee and Alexander Gray. Fast high-dimensional kernel summations using the monte carlo multipole method. In *Advances in Neural Information Processing Systems 21*, 2008.

[16] D. D. Lewis, Y. Yang, T. Rose, and F. Li. RCV1: A new benchmark collection for text categorization research. *Journal of Machine Learning Research*, 2004.

[17] Ting Liu, Andrew Moore, and Alexander Gray. New algorithms for efficient high-dimensional nonparametric classification. *Journal of Machine Learning Research*, 2006.

[18] Ting Liu, Andrew Moore, Alexander Gray, and Ke Yang. An investigation of practical approximate neighbor algorithms. In *Advances in Neural Information Processing Systems*, 2004.

[19] Frank Nielsen, Jean-Daniel Boissonnat, and Richard Nock. On bregman voronoi diagrams. In *Symposium on Discrete Algorithms*, pages 746–755, 2007.

[20] Frank Nielsen, Paolo Piro, and Michel Barlaud. Bregman vantage point trees for efficient nearest neighbor queries. In *IEEE International Conference on Multimedia & Expo*, 2009.

[21] Frank Nielsen, Paolo Piro, and Michel Barlaud. Tailored bregman ball trees for effective nearest neighbors. In *European Workshop on Computational Geometry*, 2009.

[22] Fernando Pereira, Naftali Tishby, and Lillian Lee. Distributional clustering of English words. In *31st Annual Meeting of the ACL*, pages 183–190, 1993.

[23] Jan Puzicha, Joachim Buhmann, Yossi Rubner, and Carlo Tomasi. Empirical evaluation of dissimilarity measures for color and texture. In *Proceedings of the Internation Conference on Computer Vision (ICCV)*, 1999.

[24] N. Rasiwasia, P. Moreno, and N. Vasconcelos. Bridging the gap: query by semantic example. *IEEE Transactions on Multimedia*, 2007.

[25] Yirong Shen, Andrew Ng, and Matthias Seeger. Fast gaussian process regression using kd-trees. In *Advances in Neural Information Processing Systems*, 2006.

[26] Zhenjie Zhang, Beng Chin Ooi, Srinivasan Parthasarathy, and Anthony Tung. Similarity search on bregman divergence: towards non-metric indexing. In *International Conference on Very Large Databases (VLDB)*, 2009.

